# $t$-Logistic Regression

**Nan Ding[2], S.V. N. Vishwanathan[1,2]**
Departments of [1]Statistics and [2]Computer Science
Purdue University
ding10@purdue.edu, vishy@stat.purdue.edu

## Abstract

We extend logistic regression by using $t$-exponential families which were intro- duced recently in statistical physics. This gives rise to a regularized risk mini- mization problem with a non-convex loss function. An efficient block coordinate descent optimization scheme can be derived for estimating the parameters. Be- cause of the nature of the loss function, our algorithm is tolerant to label noise. Furthermore, unlike other algorithms which employ non-convex loss functions, our algorithm is fairly robust to the choice of initial values. We verify both these observations empirically on a number of synthetic and real datasets.

## 1 Introduction

Many machine learning algorithms minimize a regularized risk [1]:

$$J(\boldsymbol{\theta}) = \Omega(\boldsymbol{\theta}) + \mathrm{R_{emp}}(\boldsymbol{\theta}), \text{ where } \mathrm{R_{emp}}(\boldsymbol{\theta}) = \frac{1}{m}\sum_{i=1}^{m} l(\mathbf{x}_i, y_i, \boldsymbol{\theta}). \tag{1}$$

Here, $\Omega$ is a regularizer which penalizes complex $\boldsymbol{\theta}$; and $\mathrm{R_{emp}}$, the empirical risk, is obtained by averaging the loss $l$ over the training dataset $\{(\mathbf{x}_1, y_1), \ldots, (\mathbf{x}_m, y_m)\}$. In this paper our focus is on binary classification, wherein features of a data point $\mathbf{x}$ are extracted via a feature map $\phi$ and the label is usually predicted via $\mathrm{sign}(\langle \phi(\mathbf{x}), \boldsymbol{\theta}\rangle)$. If we define the *margin* of a training example $(\mathbf{x}, y)$ as $u(\mathbf{x}, y, \boldsymbol{\theta}) := y\langle \phi(\mathbf{x}), \boldsymbol{\theta}\rangle$, then many popular loss functions for binary classification can be written as functions of the margin. Examples include[1]

$$l(u) = 0 \text{ if } u > 0 \text{ and } 1 \text{ otherwise .} \qquad (0-1 \text{ loss}) \tag{2}$$
$$l(u) = \max(0, 1-u) \qquad (\text{Hinge Loss}) \tag{3}$$
$$l(u) = \exp(-u) \qquad (\text{Exponential Loss}) \tag{4}$$
$$l(u) = \log(1 + \exp(-u)) \qquad (\text{Logistic Loss}). \tag{5}$$

The $0-1$ loss is non-convex and difficult to handle; it has been shown that it is NP-hard to even approximately minimize the regularized risk with the $0-1$ loss [2]. Therefore, other loss functions can be viewed as convex proxies of the $0-1$ loss. Hinge loss leads to support vector machines (SVMs), exponential loss is used in Adaboost, and logistic regression uses the logistic loss.

Convexity is a very attractive property because it ensures that the regularized risk minimization problem has a unique global optimum [3]. However, as was recently shown by Long and Servedio [4], learning algorithms based on convex loss functions are not robust to noise[2]. Intuitively, the convex loss functions grows at least linearly with slope $|l'(0)|$ as $u \in (-\infty, 0)$, which introduces the overwhelming impact from the data with $u \ll 0$. There has been some recent and some not- so-recent work on using non-convex loss functions to alleviate the above problem. For instance, a recent manuscript by [5] uses the cdf of the Guassian distribution to define a non-convex loss.

In this paper, we continue this line of inquiry and propose a non-convex loss function which is firmly grounded in probability theory. By extending logistic regression from the exponential family to the $t$-exponential family, a natural extension of exponential family of distributions studied in statistical physics [6–10], we obtain the $t$-logistic regression algorithm. Furthermore, we show that a simple block coordinate descent scheme can be used to solve the resultant regularized risk minimization problem. Analysis of this procedure also intuitively explains why $t$-logistic regression is able to handle label noise.

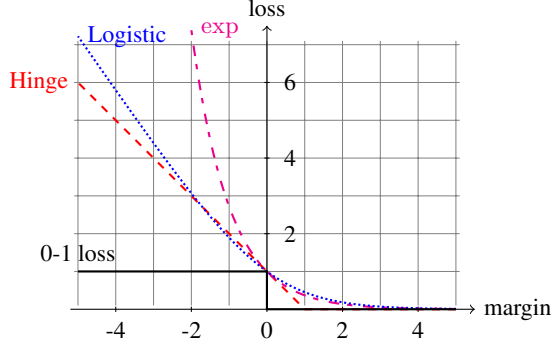

**Figure 1:** Some commonly used loss functions for binary classification. The 0-1 loss is non-convex. The hinge, exponential, and logistic losses are convex upper bounds of the 0-1 loss.

Our paper is structured as follows: In section 2 we briefly review logistic regression especially in the context of exponential families. In section 3, we review $t$-exponential families, which form the basis for our proposed $t$-logistic regression algorithm introduced in section 4. In section 5 we utilize ideas from convex multiplicative programming to design an optimization strategy. Experiments that compare our new approach to existing algorithms on a number of publicly available datasets are reported in section 6, and the paper concludes with a discussion and outlook in section 7. Some technical details as well as extra experimental results can be found in the supplementary material.

## 2   Logistic Regression

Since we build upon the probabilistic underpinnings of logistic regression, we briefly review some salient concepts. Details can be found in any standard textbook such as [11] or [12]. Assume we are given a labeled dataset $(\mathbf{X}, \mathbf{Y}) = \{(\mathbf{x}_1, y_1), \ldots, (\mathbf{x}_m, y_m)\}$ with the $\mathbf{x}_i$'s drawn from some domain $\mathcal{X}$ and the labels $y_i \in \{\pm 1\}$. Given a family of conditional distributions parameterized by $\boldsymbol{\theta}$, using Bayes rule, and making a standard iid assumption about the data allows us to write

$$p(\boldsymbol{\theta} \,|\, \mathbf{X}, \mathbf{Y}) = p(\boldsymbol{\theta}) \prod_{i=1}^{m} p(y_i \,|\, \mathbf{x}_i; \boldsymbol{\theta}) / p(\mathbf{Y} \,|\, \mathbf{X}) \propto p(\boldsymbol{\theta}) \prod_{i=1}^{m} p(y_i \,|\, \mathbf{x}_i; \boldsymbol{\theta}) \tag{6}$$

where $p(\mathbf{Y} \,|\, \mathbf{X})$ is clearly independent of $\boldsymbol{\theta}$. To model $p(y_i \,|\, \mathbf{x}_i; \boldsymbol{\theta})$, consider the conditional exponential family of distributions

$$p(y \,|\, \mathbf{x}; \boldsymbol{\theta}) = \exp\left(\langle \phi(\mathbf{x}, y), \boldsymbol{\theta} \rangle - g(\boldsymbol{\theta} \,|\, \mathbf{x})\right), \tag{7}$$

with the log-partition function $g(\boldsymbol{\theta} \,|\, \mathbf{x})$ given by

$$g(\boldsymbol{\theta} \,|\, \mathbf{x}) = \log\left(\exp\left(\langle \phi(\mathbf{x}, +1), \boldsymbol{\theta} \rangle\right) + \exp\left(\langle \phi(\mathbf{x}, -1), \boldsymbol{\theta} \rangle\right)\right). \tag{8}$$

If we choose the feature map $\phi(\mathbf{x}, y) = \frac{y}{2} \phi(\mathbf{x})$, and denote $u = y \langle \phi(\mathbf{x}), \boldsymbol{\theta} \rangle$ then it is easy to see that $p(y \,|\, \mathbf{x}; \boldsymbol{\theta})$ is the logistic function

$$p(y \,|\, \mathbf{x}; \boldsymbol{\theta}) = \frac{\exp(u/2)}{\exp(u/2) + \exp(-u/2)} = \frac{1}{1 + \exp(-u)}. \tag{9}$$

By assuming a zero mean isotropic Gaussian prior $\mathcal{N}(0, \frac{1}{\sqrt{\lambda}} I)$ for $\boldsymbol{\theta}$, plugging in (9), and taking logarithms, we can rewrite (6) as

$$-\log p(\boldsymbol{\theta} \,|\, \mathbf{X}, \mathbf{Y}) = \frac{\lambda}{2} \|\boldsymbol{\theta}\|^2 + \sum_{i=1}^{m} \log\left(1 + \exp\left(-y_i \langle \phi(\mathbf{x}_i), \boldsymbol{\theta} \rangle\right)\right) + \text{const.} \ . \tag{10}$$

Logistic regression computes a maximum a-posteriori (MAP) estimate for $\boldsymbol{\theta}$ by minimizing (10) as a function of $\boldsymbol{\theta}$. Comparing (1) and (10) it is easy to see that the regularizer employed in logistic regression is $\frac{\lambda}{2} \|\boldsymbol{\theta}\|^2$, while the loss function is the negative log-likelihood $-\log p(y \,|\, \mathbf{x}; \boldsymbol{\theta})$, which thanks to (9) can be identified with the logistic loss (5).

# 3  $t$-Exponential family of Distributions

In this section we will look at generalizations of the $\log$ and $\exp$ functions which were first introduced in statistical physics [6–9]. Some extensions and machine learning applications were presented in [13]. In fact, a more general class of functions was studied in these publications, but for our purposes we will restrict our attention to the so-called $t$-exponential and $t$-logarithm functions.

The $t$-exponential function $\exp_t$ for $(0 < t < 2)$ is defined as follows:

$$\exp_t(x) := \begin{cases} \exp(x) & \text{if } t = 1 \\ [1 + (1-t)x]_+^{1/(1-t)} & \text{otherwise.} \end{cases} \tag{11}$$

where $(\cdot)_+ = \max(\cdot, 0)$. Some examples are shown in Figure 2. Clearly, $\exp_t$ generalizes the usual $\exp$ function, which is recovered in the limit as $t \to 1$. Furthermore, many familiar properties of $\exp$ are preserved: $\exp_t$ functions are convex, non-decreasing, non-negative and satisfy $\exp_t(0) = 1$ [9]. But $\exp_t$ does not preserve one very important property of $\exp$, namely $\exp_t(a + b) \neq \exp_t(a) \cdot \exp_t(b)$. One can also define the inverse of $\exp_t$ namely $\log_t$ as

$$\log_t(x) := \begin{cases} \log(x) & \text{if } t = 1 \\ \left(x^{1-t} - 1\right)/(1-t) & \text{otherwise.} \end{cases} \tag{12}$$

Similarly, $\log_t(ab) \neq \log_t(a) + \log_t(b)$. From Figure 2, it is clear that $\exp_t$ decays towards 0 more slowly than the $\exp$ function for $1 < t < 2$. This important property leads to a family of heavy tailed distributions which we will later exploit.

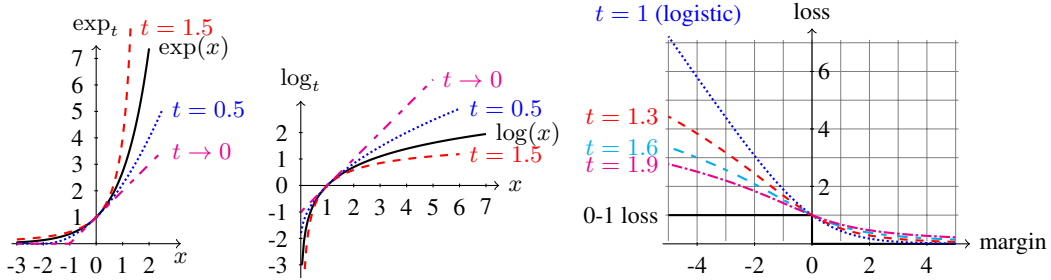

Figure 2: Left: $\exp_t$ and Middle: $\log_t$ for various values of $t$ indicated. The right figure depicts the $t$-logistic loss functions for different values of $t$. When $t = 1$, we recover the logistic loss

Analogous to the exponential family of distributions, the $t$-exponential family of distributions is defined as [9, 13]:

$$p(\mathbf{x}; \boldsymbol{\theta}) := \exp_t\left(\langle \phi(\mathbf{x}), \boldsymbol{\theta} \rangle - g_t(\boldsymbol{\theta})\right). \tag{13}$$

A prominent member of the $t$-exponential family is the Student's-$t$ distribution [14]. Just like in the exponential family case, $g_t$ the log-partition function ensures that $p(\mathbf{x}; \boldsymbol{\theta})$ is normalized. However, no closed form solution exists for computing $g_t$ exactly in general. A closely related distribution, which often appears when working with $t$-exponential families is the so-called escort distribution [9, 13]:

$$q_t(x; \boldsymbol{\theta}) := p(x; \boldsymbol{\theta})^t / Z(\boldsymbol{\theta}) \tag{14}$$

where $Z(\boldsymbol{\theta}) = \int p(x; \boldsymbol{\theta})^t dx$ is the normalizing constant which ensures that the escort distribution integrates to 1.

Although $g_t(\boldsymbol{\theta})$ is not the cumulant function of the $t$-exponential family, it still preserves convexity. In addition, it is very close to being a moment generating function

$$\nabla_{\boldsymbol{\theta}} g_t(\boldsymbol{\theta}) = \mathbb{E}_{q_t(\mathbf{x}; \boldsymbol{\theta})}\left[\phi(\mathbf{x})\right]. \tag{15}$$

The proof is provided in the supplementary material. A general version of this result appears as Lemma 3.8 in Sears [13] and a version specialized to the generalized exponential families appears as Proposition 5.2 in [9]. The main difference from $\nabla_{\boldsymbol{\theta}} g(\boldsymbol{\theta})$ of the normal exponential family is that now $\nabla_{\boldsymbol{\theta}} g_t(\boldsymbol{\theta})$ is equal to the expectation of its escort distribution $q_t(x; \boldsymbol{\theta})$ instead of $p(x; \boldsymbol{\theta})$.

# 4 Binary Classification with the $t$-exponential Family

In $t$-logistic regression we model $p(y|\mathbf{x};\boldsymbol{\theta})$ via a conditional $t$-exponential family distribution

$$p(y|\mathbf{x};\boldsymbol{\theta}) = \exp_t\left(\langle\phi(\mathbf{x},y),\boldsymbol{\theta}\rangle - g_t(\boldsymbol{\theta}\,|\,\mathbf{x})\right), \tag{16}$$

where $1 < t < 2$, and compute the log-partition function $g_t$ by noting that

$$\exp_t\left(\langle\phi(\mathbf{x},+1),\boldsymbol{\theta}\rangle - g_t(\boldsymbol{\theta}\,|\,\mathbf{x})\right) + \exp_t\left(\langle\phi(\mathbf{x},-1),\boldsymbol{\theta}\rangle - g_t(\boldsymbol{\theta}\,|\,\mathbf{x})\right) = 1. \tag{17}$$

Even though no closed form solution exists, one can compute $g_t$ given $\boldsymbol{\theta}$ and $\mathbf{x}$ using numerical techniques efficiently.

The Student's-$t$ distribution can be regarded as a counterpart of the isotropic Gaussian prior in the $t$-exponential family [14]. Recall that a one dimensional Student's-$t$ distribution is given by

$$St(x|\mu,\sigma,v) = \frac{\Gamma((v+1)/2)}{\sqrt{v\pi}\Gamma(v/2)\sigma^{1/2}}\left(1 + \frac{(x-\mu)^2}{v\sigma}\right)^{-(v+1)/2}, \tag{18}$$

where $\Gamma(\cdot)$ denotes the usual Gamma function and $v > 1$ so that the mean is finite. If we select $t$ satisfying $-(v+1)/2 = 1/(1-t)$ and denote,

$$\Psi = \left(\frac{\Gamma((v+1)/2)}{\sqrt{v\pi}\Gamma(v/2)\sigma^{1/2}}\right)^{-2/(v+1)},$$

then by some simple but tedious calculation (included in the supplementary material)

$$St(x|\mu,\sigma,v) = \exp_t(-\tilde{\lambda}(x-\mu)^2/2 - \tilde{g}_t) \tag{19}$$

$$\text{where}\quad \tilde{\lambda} = \frac{2\Psi}{(t-1)v\sigma}\quad\text{and}\quad \tilde{g}_t = \frac{\Psi-1}{t-1}.$$

Therefore, we work with the Student's-$t$ prior in our setting:

$$p(\boldsymbol{\theta}) = \prod_{j=1}^{d} p(\theta_j) = \prod_{j=1}^{d} St(\theta_j|0,2/\lambda,(3-t)/(t-1)). \tag{20}$$

Here, the degree of freedom for Student's-$t$ distribution is chosen such that it also belongs to the $\exp_t$ family, which in turn yields $v = (3-t)/(t-1)$. The Student's-$t$ prior is usually preferred to the Gaussian prior when the underlying distribution is heavy-tailed. In practice, it is known to be a robust[3] alternative to the Gaussian distribution [16, 17].

As before, if we let $\phi(\mathbf{x},y) = \frac{y}{2}\phi(\mathbf{x})$ and plot the negative log-likelihood $-\log p(y|\mathbf{x};\boldsymbol{\theta})$, then we no longer obtain a convex loss function (see Figure 2). Similarly, $-\log p(\boldsymbol{\theta})$ is no longer convex when we use the Student's-$t$ prior. This makes optimizing the regularized risk challenging, therefore we employ a different strategy.

Since $\log_t$ is also a monotonically increasing function, instead of working with $\log$, we can equivalently work with the $\log_t$ function (12) and minimize the following objective function:

$$\hat{J}(\boldsymbol{\theta}) = -\log_t p(\boldsymbol{\theta})\prod_{i=1}^{m} p(y_i|\mathbf{x}_i;\boldsymbol{\theta})/p(\mathbf{Y}\,|\,\mathbf{X})$$

$$= \frac{1}{t-1}\left(p(\boldsymbol{\theta})\prod_{i=1}^{m} p(y_i|\mathbf{x}_i;\boldsymbol{\theta})/p(\mathbf{Y}\,|\,\mathbf{X})\right)^{1-t} + \frac{1}{1-t}, \tag{21}$$

where $p(\mathbf{Y}\,|\,\mathbf{X})$ is independent of $\boldsymbol{\theta}$. Using (13), (18), and (11), we can further write

$$\hat{J}(\boldsymbol{\theta}) \propto \prod_{j=1}^{d}\underbrace{\left(1+(1-t)(-\tilde{\lambda}\theta_j^2/2-\tilde{g}_t)\right)}_{r_j(\boldsymbol{\theta})}\prod_{i=1}^{m}\underbrace{\left(1+(1-t)(\langle\frac{y_i}{2}\phi(\mathbf{x}_i),\boldsymbol{\theta}\rangle - g_t(\boldsymbol{\theta}\,|\,\mathbf{x}_i))\right)}_{l_i(\boldsymbol{\theta})} + \text{const.}.$$

$$= \prod_{j=1}^{d} r_j(\boldsymbol{\theta})\prod_{i=1}^{m} l_i(\boldsymbol{\theta}) + \text{const.} \tag{22}$$

Since $t > 1$, it is easy to see that $r_j(\boldsymbol{\theta}) > 0$ is a convex function of $\boldsymbol{\theta}$. On the other hand, since $g_t$ is convex and $t > 1$ it follows that $l_i(\boldsymbol{\theta}) > 0$ is also a convex function of $\boldsymbol{\theta}$. In summary, $\hat{J}(\boldsymbol{\theta})$ is a product of positive convex functions. In the next section we will present an efficient optimization strategy for dealing with such problems.

## 5   Convex Multiplicative Programming

In convex multiplicative programming [18] we are interested in the following optimization problem:

$$\min_{\boldsymbol{\theta}} \;\; P(\boldsymbol{\theta}) \triangleq \prod_{n=1}^{N} z_n(\boldsymbol{\theta}) \;\; \text{s.t.} \;\; \boldsymbol{\theta} \in \mathbb{R}^d, \tag{23}$$

where $z_n(\boldsymbol{\theta})$ are positive convex functions. Clearly, (22) can be identified with (23) by setting $N = d + m$ and identifying $z_n(\boldsymbol{\theta}) = r_n(\boldsymbol{\theta})$ for $n = 1, \dots, d$ and $z_{n+d}(\boldsymbol{\theta}) = l_n(\boldsymbol{\theta})$ for $n = 1, \dots, m$.

The optimal solutions to the problem (23) can be obtained by solving the following parametric problem (see Theorem 2.1 of Kuno et al. [18]):

$$\min_{\boldsymbol{\xi}} \min_{\boldsymbol{\theta}} \;\; MP(\boldsymbol{\theta}, \boldsymbol{\xi}) \triangleq \sum_{n=1}^{N} \xi_n z_n(\boldsymbol{\theta}) \;\; \text{s.t.} \;\; \boldsymbol{\theta} \in \mathbb{R}^d, \;\; \boldsymbol{\xi} > 0, \;\; \prod_{n=1}^{N} \xi_n \geq 1. \tag{24}$$

The optimization problem in (24) is very reminiscent of logistic regression. In logistic regression, $l_n(\boldsymbol{\theta}) = -\left\langle \frac{y_n}{2}\phi(\mathbf{x}_n), \boldsymbol{\theta} \right\rangle + g(\boldsymbol{\theta} \mid \mathbf{x}_n)$, while here $l_n(\boldsymbol{\theta}) = 1 + (1-t)\left(\left\langle \frac{y_n}{2}\phi(\mathbf{x}_n), \boldsymbol{\theta} \right\rangle - g_t(\boldsymbol{\theta} \mid \mathbf{x}_n)\right)$. The key difference is that in $t$-logistic regression each data point $\mathbf{x}_n$ has a weight (or influence) $\xi_n$ associated with it.

Exact algorithms have been proposed for solving (24) (for instance, [18]). However, the computational cost of these algorithms grows exponentially with respect to $N$ which makes them impractical for our purposes. Instead, we apply a block coordinate descent based method. The main idea is to minimize (24) with respect to $\boldsymbol{\theta}$ and $\boldsymbol{\xi}$ separately.

$\boldsymbol{\xi}$-**Step**: Assume that $\boldsymbol{\theta}$ is fixed, and denote $\tilde{z}_n = z_n(\boldsymbol{\theta})$ to rewrite (24) as:

$$\min_{\boldsymbol{\xi}} \;\; \sum_{n=1}^{N} \xi_n \tilde{z}_n \;\; \text{s.t.} \;\; \boldsymbol{\xi} > 0, \;\; \prod_{n=1}^{N} \xi_n \geq 1. \tag{25}$$

Since the objective function is linear in $\boldsymbol{\xi}$ and the feasible region is a convex set, (25) is a convex optimization problem. By introducing a non-negative Lagrange multiplier $\gamma \geq 0$, the partial Lagrangian and its gradient with respect to $\xi_{n'}$ can be written as

$$L(\boldsymbol{\xi}, \gamma) = \sum_{n=1}^{N} \xi_n \tilde{z}_n + \gamma \cdot \left(1 - \prod_{n=1}^{N} \xi_n\right) \tag{26}$$

$$\frac{\partial}{\partial \xi_{n'}} L(\xi, \gamma) = \tilde{z}_{n'} - \gamma \prod_{n \neq n'} \xi_n. \tag{27}$$

Setting the gradient to 0 obtains $\gamma = \frac{\tilde{z}_{n'}}{\prod_{n \neq n'} \xi_n}$. Since $\tilde{z}_{n'} > 0$, it follows that $\gamma$ cannot be 0. By the K.K.T. conditions [3], we can conclude that $\prod_{n=1}^{N} \xi_n = 1$. This in turn implies that $\gamma = \tilde{z}_{n'} \xi_{n'}$ or

$$(\xi_1, \dots, \xi_N) = (\gamma/\tilde{z}_1, \dots, \gamma/\tilde{z}_N), \;\; \text{with} \;\; \gamma = \prod_{n=1}^{N} \tilde{z}_n^{\frac{1}{N}}. \tag{28}$$

Recall that $\xi_n$ in (24) is the weight (or influence) of each term $z_n(\boldsymbol{\theta})$. The above analysis shows that $\gamma = \tilde{z}_n(\boldsymbol{\theta})\xi_n$ remains constant for all $n$. If $\tilde{z}_n(\boldsymbol{\theta})$ becomes very large then its influence $\xi_n$ is reduced. Therefore, points with very large loss have their influence capped and this makes the algorithm robust to outliers.

$\boldsymbol{\theta}$-**Step**: In this step we fix $\boldsymbol{\xi} > 0$ and solve for the optimal $\boldsymbol{\theta}$. This step is essentially the same as logistic regression, except that each component has a weight $\xi$ here.

$$\min_{\boldsymbol{\theta}} \;\; \sum_{n=1}^{N} \xi_n z_n(\boldsymbol{\theta}) \;\; \text{s.t.} \;\; \boldsymbol{\theta} \in \mathbb{R}^d. \tag{29}$$

This is a standard unconstrained convex optimization problem which can be solved by any off the shelf solver. In our case we use the L-BFGS Quasi-Newton method. This requires us to compute the gradient $\nabla_{\boldsymbol{\theta}} z_n(\boldsymbol{\theta})$:

$$\text{for } n = 1, \ldots, d \quad \nabla_{\boldsymbol{\theta}} z_n(\boldsymbol{\theta}) = \nabla_{\boldsymbol{\theta}} r_n(\boldsymbol{\theta}) = (t-1)\tilde{\lambda}\theta_n \cdot \mathbf{e}_n$$

$$\text{for } n = 1, \ldots, m \quad \nabla_{\boldsymbol{\theta}} z_{n+d}(\boldsymbol{\theta}) = \nabla_{\boldsymbol{\theta}} l_n(\boldsymbol{\theta}) = (1-t)\left(\frac{y_n}{2}\phi(\mathbf{x}_n) - \nabla_{\boldsymbol{\theta}} g_t(\boldsymbol{\theta} \,|\, \mathbf{x}_n)\right)$$

$$= (1-t)\left(\frac{y_n}{2}\phi(\mathbf{x}_n) - \mathbb{E}_{q_t(y_n|\mathbf{x}_n;\boldsymbol{\theta})}\left[\frac{y_n}{2}\phi(\mathbf{x}_n)\right]\right),$$

where $\mathbf{e}_n$ denotes the $d$ dimensional vector with one at the $n$-th coordinate and zeros elsewhere ($n$-th unit vector). $q_t(y|\mathbf{x};\boldsymbol{\theta})$ is the escort distribution of $p(y|\mathbf{x};\boldsymbol{\theta})$ (16):

$$q_t(y|\mathbf{x};\boldsymbol{\theta}) = \frac{p(y|\mathbf{x};\boldsymbol{\theta})^t}{p(+1|\mathbf{x};\boldsymbol{\theta})^t + p(-1|\mathbf{x};\boldsymbol{\theta})^t}. \qquad (30)$$

The objective function is monotonically decreasing and is guaranteed to converge to a stable point of $\mathrm{P}(\theta)$. We include the proof in the supplementary material.

## 6  Experimental Evaluation

Our experimental evaluation is designed to answer four natural questions: 1) How does the generalization capability (measured in terms of test error) of $t$-logistic regression compare with existing algorithms such as logistic regression and support vector machines (SVMs) both in the presence and absence of label noise? 2) Do the $\xi$ variables we introduced in the previous section have a natural interpretation? 3) How much overhead does $t$-logistic regression incur as compared to logistic regression? 4) How sensitive is the algorithm to initialization? The last question is particularly important given that the algorithm is minimizing a non-convex loss.

To answer the above questions empirically we use six datasets, two of which are synthetic. The Long-Servedio dataset is an artificially constructed dataset to show that algorithms which minimize a differentiable convex loss are not tolerant to label noise Long and Servedio [4]. The examples have 21 dimensions and play one of three possible roles: **large margin examples** (25%, $x_{1,2,\ldots,21} = y$); **pullers** (25%, $x_{1,\ldots,11} = y$, $x_{12,\ldots,21} = -y$); and **penalizers** (50%, Randomly select and set 5 of the first 11 coordinates and 6 out of the last 10 coordinates to $y$, and set the remaining coordinates to $-y$). The Mease-Wyner is another synthetic dataset to test the effect of label noise. The input $x$ is a 20-dimensional vector where each coordinate is uniformly distributed on $[0, 1]$. The label $y$ is $+1$ if $\sum_{j=1}^{5} x_j \geq 2.5$ and $-1$ otherwise [19]. In addition, we also test on Mushroom, USPS-N (9 vs. others), Adult, and Web datasets, which are often used to evaluate machine learning algorithms (see Table 1 in supplementary material for details).

For simplicity, we use the identity feature map $\phi(\mathbf{x}) = \mathbf{x}$ in all our experiments, and set $t \in \{1.3, 1.6, 1.9\}$ for $t$-logistic regression. Our comparators are logistic regression, linear SVMs[4], and an algorithm (the probit) which employs the probit loss, $\mathcal{L}(u) = 1 - erf(2u)$, used in Brown-Boost/RobustBoost [5]. We use the L-BFGS algorithm [21] for the $\boldsymbol{\theta}$-step in $t$-logistic regression. L-BFGS is also used to train logistic regression and the probit loss based algorithms. Label noise is added by randomly choosing 10% of the labels in the training set and flipping them; each dataset is tested with and without label noise. We randomly select and hold out 30% of each dataset as a validation set and use the rest of the 70% for 10-fold cross validation. The optimal parameters namely $\lambda$ for $t$-logistic and logistic regression and $C$ for SVMs is chosen by performing a grid search over the parameter space $\{2^{-7, -6, \ldots, 7}\}$ and observing the prediction accuracy over the validation set. The convergence criterion is to stop when the change in the objective function value is less than $10^{-4}$. All code is written in Matlab, and for the linear SVM we use the Matlab interface of LibSVM [22]. Experiments were performed on a Qual-core machine with Dual 2.5 Ghz processor and 32 Gb RAM.

In Figure 3, we plot the test error with and without label noise. In the latter case, the test error of $t$-logistic regression is very similar to logistic regression and Linear SVM (with 0% test error in

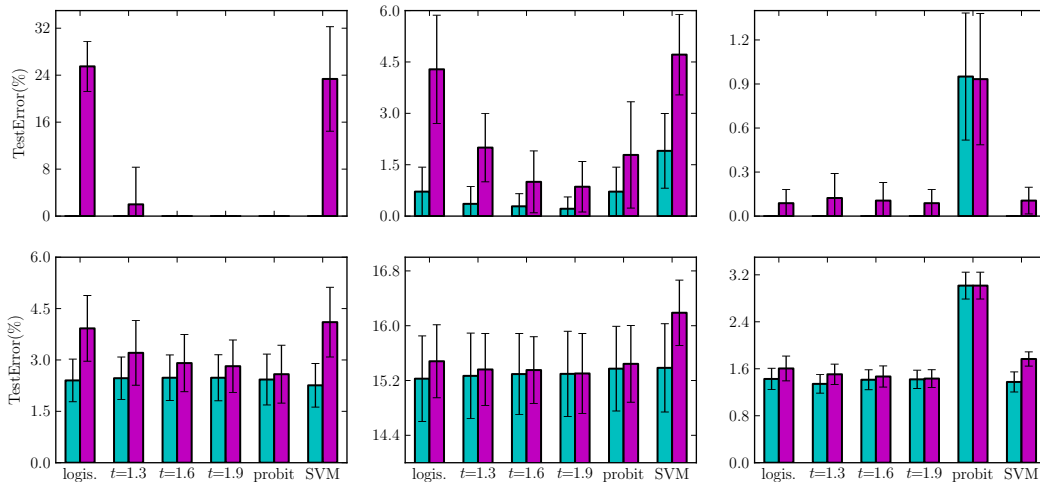

Figure 3: The test error rate of various algorithms on six datasets (left to right, top: Long-Servedio, Mease-Wyner, Mushroom; bottom: USPS-N, Adult, Web) with and without 10% label noise. All algorithms are initialized with $\boldsymbol{\theta} = 0$. The blue (light) bar denotes a clean dataset while the magenta (dark) bar are the results with label noise added. Also see Table 3 in the supplementary material.

Long-Servedio and Mushroom datasets), with a slight edge on some datasets such as Mease-Wyner. When label noise is added, $t$-logistic regression (especially with $t = 1.9$) shows significantly[5] better performance than all the other algorithms on all datasets except the USPS-N, where it is marginally outperformed by the probit.

To obtain Figure 4 we used the noisy version of the datasets, chose one of the 10 folds used in the previous experiment, and plotted the distribution of the $1/z \propto \xi$ obtained after training with $t = 1.9$. To distinguish the points with noisy labels we plot them in cyan while the other points are plotted in red. Analogous plots for other values of $t$ can be found in the supplementary material. Recall that $\xi$ denotes the influence of a point. One can clearly observe that the $\xi$ of the noisy data is much smaller than that of the clean data, which indicates that the algorithm is able to effectively identify these points and cap their influence. In particular, on the Long-Servedio dataset observe the 4 distinct spikes. From left to right, the first spike corresponds to the noisy large margin examples, the second spike represents the noisy pullers, the third spike denotes the clean pullers, while the rightmost spike corresponds to the clean large margin examples. Clearly, the noisy large margin examples and the noisy pullers are assigned a low value of $\xi$ thus capping their influence and leading to the perfect classification of the test set. On the other hand, logistic regression is unable to discriminate between clean and noisy training samples which leads to bad performance on noisy datasets.

Detailed timing experiments can be found in Table 4 in the supplementary material. In a nutshell, $t$-logistic regression takes longer to train than either logistic regression or the probit. The reasons are not difficult to see. First, there is no closed form expression for $g_t(\boldsymbol{\theta} \mid \mathbf{x})$. We therefore resort to pre-computing it at some fixed locations and using a spline method to interpolate values at other locations. Second, since the objective function is not convex several iterations of the $\boldsymbol{\xi}$ and $\boldsymbol{\theta}$ steps might be needed. Surprisingly, the L-BFGS algorithm, which is not designed to optimize non-convex functions, is able to minimize (22) directly in many cases. When it does converge, it is often faster than the convex multiplicative programming algorithm. However, on some cases (as expected) it fails to find a direction of descent and exits. A common remedy for this is the bundle L-BFGS with a trust-region approach. [21]

Given that the $t$-logistic objective function is non-convex, one naturally worries about how different initial values affect the quality of the final solution. To answer this question, we initialized the algorithm with 50 different randomly chosen $\boldsymbol{\theta} \in [-0.5, 0.5]^d$, and report test performances of the various solutions obtained in Figure 5. Just like logistic regression which uses a convex loss and hence converges to the same solution independent of the initialization, the solution obtained

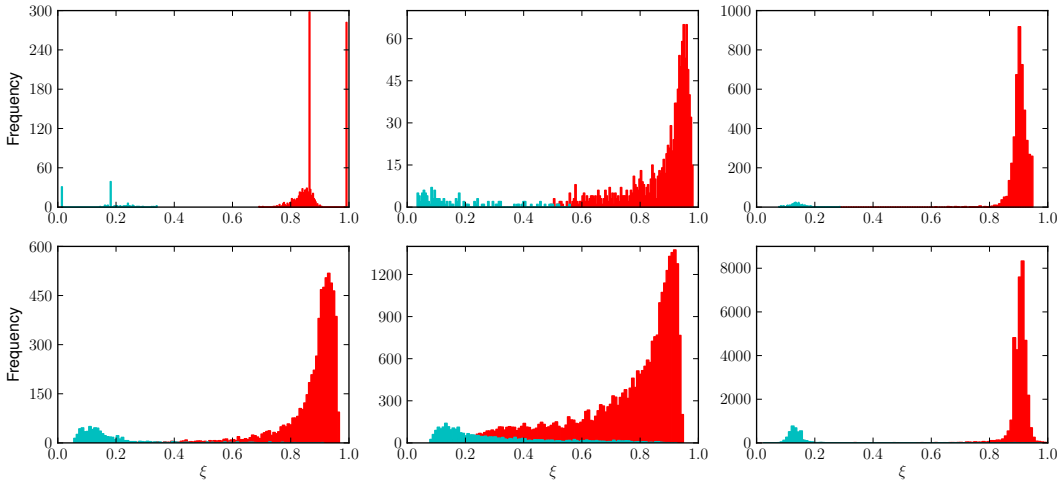

Figure 4: The distribution of $\xi$ obtained after training $t$-logistic regression with $t = 1.9$ on datasets with 10% label noise. Left to right, top: Long-Servedio, Mease-Wyner, Mushroom; bottom: USPS-N, Adult, Web. The red (dark) bars (resp. cyan (light) bars) indicate the frequency of $\xi$ assigned to points without (resp. with) label noise.

by $t$-logistic regression seems fairly independent of the initial value of $\boldsymbol{\theta}$. On the other hand, the performance of the probit fluctuates widely with different initial values of $\boldsymbol{\theta}$.

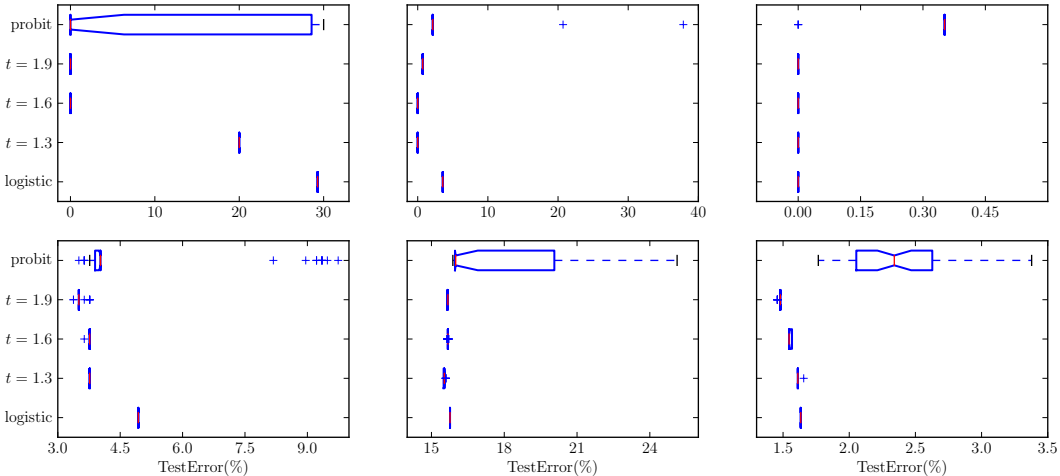

Figure 5: The Error rate by different initialization. Left to right, top: Long-Servedio, Mease-Wyner, Mushroom; bottom: USPS-N, Adult, Web.

# 7 Discussion and Outlook

In this paper, we generalize logistic regression to $t$-logistic regression by using the $t$-exponential family. The new algorithm has a probabilistic interpretation and is more robust to label noise. Even though the resulting objective function is non-convex, empirically it appears to be insensitive to initialization. There are a number of avenues for future work. On Long-Servedio experiment, if the label noise is increased significantly beyond 10%, the performance of $t$-logistic regression may degrade (see Fig. 6 in supplementary materials). Understanding and explaining this issue theoretically and empirically remains an open problem. It will be interesting to investigate if $t$-logistic regression can be married with graphical models to yield $t$-conditional random fields. We will also focus on better numerical techniques to accelerate the $\boldsymbol{\theta}$-step, especially a faster way to compute $g_t$.

## Footnotes

[1]We slightly abuse notation and use $l(u)$ to denote $l(u(\mathbf{x}, y, \boldsymbol{\theta}))$.

[2]Although, the analysis of [4] is carried out in the context of boosting, we believe, the results hold for a larger class of algorithms which minimize a regularized risk with a convex loss function.

[3]There is no unique definition of robustness. For example, one of the definitions is through the outlier-proneness [15]: $p(\boldsymbol{\theta}\,|\,\mathbf{X},\mathbf{Y},\mathbf{x}_{n+1},y_{n+1}) \to p(\boldsymbol{\theta}\,|\,\mathbf{X},\mathbf{Y})$ as $\mathbf{x}_{n+1} \to \infty$.

[4]We also experimented with RampSVM [20], however, the results are worser than the other algorithms. We therefore report these results in the supplementary material.

[5]We provide the significance test results in Table 2 of supplementary material.

# References

[1] Choon Hui Teo, S. V. N. Vishwanthan, Alex J. Smola, and Quoc V. Le. Bundle methods for regularized risk minimization. *J. Mach. Learn. Res.*, 11:311–365, January 2010.

[2] S. Ben-David, N. Eiron, and P.M. Long. On the difficulty of approximately maximizing agreements. *J. Comput. System Sci.*, 66(3):496–514, 2003.

[3] S. Boyd and L. Vandenberghe. *Convex Optimization*. Cambridge University Press, Cambridge, England, 2004.

[4] Phil Long and Rocco Servedio. Random classification noise defeats all convex potential boosters. *Machine Learning Journal*, 78(3):287–304, 2010.

[5] Yoav Freund. A more robust boosting algorithm. Technical Report Arxiv/0905.2138, Arxiv, May 2009.

[6] J. Naudts. Deformed exponentials and logarithms in generalized thermostatistics. *Physica A*, 316:323–334, 2002. URL http://arxiv.org/pdf/cond-mat/0203489.

[7] J. Naudts. Generalized thermostatistics based on deformed exponential and logarithmic functions. *Physica A*, 340:32–40, 2004.

[8] J. Naudts. Generalized thermostatistics and mean-field theory. *Physica A*, 332:279–300, 2004.

[9] J. Naudts. Estimators, escort proabilities, and $\phi$-exponential families in statistical physics. *Journal of Inequalities in Pure and Applied Mathematics*, 5(4), 2004.

[10] C. Tsallis. Possible generalization of boltzmann-gibbs statistics. *J. Stat. Phys.*, 52, 1988.

[11] Christopher Bishop. *Pattern Recognition and Machine Learning*. Springer, 2006.

[12] Trevor Hastie, Robert Tibshirani, and Jerome Friedman. *The Elements of Statistical Learning*. Springer, New York, 2 edition, 2009.

[13] Timothy D. Sears. *Generalized Maximum Entropy, Convexity, and Machine Learning*. PhD thesis, Australian National University, 2008.

[14] Andre Sousa and Constantino Tsallis. Student's t- and r-distributions: Unified derivation from an entropic variational principle. *Physica A*, 236:52–57, 1994.

[15] A O'hagan. On outlier rejection phenomena in bayes inference. *Royal Statistical Society*, 41 (3):358–367, 1979.

[16] Kenneth L. Lange, Roderick J. A. Little, and Jeremy M. G. Taylor. Robust statistical modeling using the t distribution. *Journal of the American Statistical Association*, 84(408):881–896, 1989.

[17] J. Vanhatalo, P. Jylanki, and A. Vehtari. Gaussian process regression with student-t likelihood. In *Neural Information Processing System*, 2009.

[18] Takahito Kuno, Yasutoshi Yajima, and Hiroshi Konno. An outer approximation method for minimizing the product of several convex functions on a convex set. *Journal of Global Optimization*, 3(3):325–335, September 1993.

[19] David Mease and Abraham Wyner. Evidence contrary to the statistical view of boosting. *J. Mach. Learn. Res.*, 9:131–156, February 2008.

[20] R. Collobert, F.H. Sinz, J. Weston, and L. Bottou. Trading convexity for scalability. In W.W. Cohen and A. Moore, editors, *Machine Learning, Proceedings of the Twenty-Third International Conference (ICML 2006)*, pages 201–208. ACM, 2006.

[21] J. Nocedal and S. J. Wright. *Numerical Optimization*. Springer Series in Operations Research. Springer, 1999.

[22] C.C. Chang and C.J. Lin. *LIBSVM: a library for support vector machines*, 2001. Software available at http://www.csie.ntu.edu.tw/~cjlin/libsvm.

[23] Fabian Sinz. *UniverSVM: Support Vector Machine with Large Scale CCCP Functionality*, 2006. Software available at http://www.kyb.mpg.de/bs/people/fabee/universvm.html.

